# Analyzing Auditory Neurons by Learning Distance Functions

**Inna Weiner**[1]     **Tomer Hertz**[1,2]     **Israel Nelken**[2,3]     **Daphna Weinshall**[1,2]

[1]School of Computer Science and Engineering,
[2]The Center for Neural Computation, [3]Department of Neurobiology,
The Hebrew University of Jerusalem, Jerusalem, Israel, 91904
`weinerin,tomboy,daphna@cs.huji.ac.il,israel@md.huji.ac.il`

## Abstract

We present a novel approach to the characterization of complex sensory neurons. One of the main goals of characterizing sensory neurons is to characterize dimensions in stimulus space to which the neurons are highly sensitive (causing large gradients in the neural responses) or alternatively dimensions in stimulus space to which the neuronal response are invariant (defining iso-response manifolds). We formulate this problem as that of learning a geometry on stimulus space that is compatible with the neural responses: the distance between stimuli should be large when the responses they evoke are very different, and small when the responses they evoke are similar. Here we show how to successfully train such distance functions using rather limited amount of information. The data consisted of the responses of neurons in primary auditory cortex (A1) of anesthetized cats to 32 stimuli derived from natural sounds. For each neuron, a subset of all pairs of stimuli was selected such that the responses of the two stimuli in a pair were either very similar or very dissimilar. The distance function was trained to fit these constraints. The resulting distance functions generalized to predict the distances between the responses of a test stimulus and the trained stimuli.

## 1   Introduction

A major challenge in auditory neuroscience is to understand how cortical neurons represent the acoustic environment. Neural responses to complex sounds are idiosyncratic, and small perturbations in the stimuli may give rise to large changes in the responses. Furthermore, different neurons, even with similar frequency response areas, may respond very differently to the same set of stimuli. The dominant approach to the functional characterization of sensory neurons attempts to predict the response of the cortical neuron to a novel stimulus. Prediction is usually estimated from a set of known responses of a given neuron to a set of stimuli (sounds). The most popular approach computes the spectrotemporal receptive field (STRF) of each neuron, and uses this linear model to predict neuronal responses. However, STRFs have been recently shown to have low predictive power [10, 14].

In this paper we take a different approach to the characterization of auditory cortical neurons. Our approach attempts to learn the non-linear warping of stimulus space that is in-

duced by the neuronal responses. This approach is motivated by our previous observations [3] that different neurons impose different partitions of the stimulus space, which are not necessarily simply related to the spectro-temporal structure of the stimuli. More specifically, we characterize a neuron by learning a pairwise distance function over the stimulus domain that will be consistent with the similarities between the responses to different stimuli, see Section 2. Intuitively a good distance function would assign small values to pairs of stimuli that elicit a similar neuronal response, and large values to pairs of stimuli that elicit different neuronal responses.

This approach has a number of potential advantages: First, it allows us to aggregate information from a number of neurons, in order to learn a good distance function even when the number of known stimuli responses per neuron is small, which is a typical concern in the domain of neuronal characterization. Second, unlike most functional characterizations that are limited to linear or weakly non-linear models, distance learning can approximate functions that are highly non-linear. Finally, we explicitly learn a distance function on stimulus space; by examining the properties of such a function, it may be possible to determine the stimulus features that most strongly influence the responses of a cortical neuron. While this information is also implicitly incorporated into functional characterizations such as the STRF, it is much more explicit in our new formulation.

In this paper we therefore focus on two questions: (1) Can we learn distance functions over the stimulus domain for single cells using information extracted from their neuronal responses?? and (2) What is the predictive power of these cell specific distance functions when presented with novel stimuli? In order to address these questions we used extracellular recordings from 22 cells in the auditory cortex of cats in response to natural bird chirps and some modified versions of these chirps [1]. To estimate the distance between responses, we used a normalized distance measure between the peri-stimulus time histograms of the responses to the different stimuli.

Our results, described in Section 4, show that we can learn compatible distance functions on the stimulus domain with relatively low training errors. This result is interesting by itself as a possible characterization of cortical auditory neurons, a goal which eluded many previous studies [3]. Using cross validation, we measure the test error (or predictive power) of our method, and report generalization power which is significantly higher than previously reported for natural stimuli [10]. We then show that performance can be further improved by learning a distance function using information from pairs of related neurons. Finally, we show better generalization performance for wide-band stimuli as compared to narrow-band stimuli. These latter two contributions may have some interesting biological implications regarding the nature of the computations done by auditory cortical neurons.

**Related work**   Recently, considerable attention has been focused on spectrotemporal receptive fields (STRFs) as characterizations of the function of auditory cortical neurons [8, 4, 2, 11, 16].    The STRF model is appealing in several respects: it is a conceptually simple model that provides a linear description of the neuron's behavior. It can be interpreted both as providing the neuron's most efficient stimulus (in the time-frequency domain), and also as the spectro-temporal impulse response of the neuron [10, 12]. Finally, STRFs can be efficiently estimated using simple algebraic techniques.

However, while there were initial hopes that STRFs would uncover relatively complex response properties of cortical neurons, several recent reports of large sets of STRFs of cortical neurons concluded that most STRFs are somewhat too simple [5], and that STRFs are typically rather sluggish in time, therefore missing the highly precise synchronization of some cortical neurons [11]. Furthermore, when STRFs are used to predict neuronal responses to natural stimuli they often fail to predict the correct responses [10, 6]. For example, in Machens et al. only $11\%$ of the response power could be predicted by STRFs on average [10]. Similar results were also reported in [14], who found that STRF models

account for only $18 - 40\%$ (on average) of the stimulus related power in auditory cortical neural responses to dynamic random chord stimuli. Various other studies have shown that there are significant and relevant non-linearities in auditory cortical responses to natural stimuli [13, 1, 9, 10]. Using natural sounds, Bar-Yosef et. al [1] have shown that auditory neurons are extremely sensitive to small perturbations in the (natural) acoustic context. Clearly, these non-linearities cannot be sufficiently explained using linear models such as the STRF.

## 2 Formalizing the problem as a distance learning problem

Our approach is based on the idea of learning a cell-specific distance function over the space of all possible stimuli, relying on partial information extracted from the neuronal responses of the cell. The initial data consists of stimuli and the resulting neural responses. We use the neuronal responses to identify pairs of stimuli to which the neuron responded similarly and pairs to which the neuron responded very differently. These pairs can be formally described by equivalence constraints. Equivalence constraints are relations between pairs of datapoints, which indicate whether the points in the pair belong to the same category or not. We term a constraint *positive* when they points are known to originate from the same class, and *negative* belong to different classes. In this setting the goal of the algorithm is to learn a distance function that attempts to comply with the equivalence constraints.

This formalism allows us to combine information from a number of cells to improve the resulting characterization. Specifically, we combine equivalence constraints gathered from pairs of cells which have similar responses, and train a single distance function for both cells. Our results demonstrate that this approach improves prediction results of the "weaker" cell, and almost always improves the result of the "stronger" cell in each pair. Another interesting result of this formalism is the ability to classify stimuli based on the responses of the total recorded cortical cell ensemble. For some stimuli, the predictive performance based on the learned inter-stimuli distance was very good, whereas for other stimuli it was rather poor. These differences were correlated with the acoustic structure of the stimuli, partitioning them into narrowband and wideband stimuli.

## 3 Methods

**Experimental setup** Extracellular recordings were made in primary auditory cortex of nine halothane-anesthetized cats. Anesthesia was induced by ketamine and xylazine and maintained with halothane (0.25-1.5%) in 70% $N_2O$ using standard protocols authorized by the committee for animal care and ethics of the Hebrew University - Haddasah Medical School. Single neurons were recorded using metal microelectrodes and an online spike sorter (MSD, alpha-omega). All neurons were well separated. Penetrations were performed over the whole dorso-ventral extent of the appropriate frequency slab (between about 2 and 8 kHz). Stimuli were presented 20 times using sealed, calibrated earphones at 60-80 dB SPL, at the preferred aurality of the neurons as determined using broad-band noise bursts. Sounds were taken from the Cornell Laboratory of Ornithology and have been selected as in [1]. Four stimuli, each of length 60-100 ms, consisted of a main tonal component with frequency and amplitude modulation and of a background noise consisting of echoes and unrelated components. Each of these stimuli was further modified by separating the main tonal component from the noise, and by further separating the noise into echoes and background. All possible combinations of these components were used here, in addition to a stylized artificial version that lacked the amplitude modulation of the natural sound. In total, 8 versions of each stimulus were used, and therefore each neuron had a dataset consisting of 32 datapoints. For more detailed methods, see Bar-Yosef et al. [1].

**Data representation**   We used the first 60 ms of each stimulus. Each stimulus was represented using the first $d$ real Cepstral coefficients. The real Cepstrum of a signal $x$ was calculated by taking the natural logarithm of magnitude of the Fourier transform of $x$ and then computing the inverse Fourier transform of the resulting sequence. In our experiments we used the first 21-30 coefficients. Neuronal responses were represented by creating Peri-Stimulus Time Histograms (PSTHs) using 20 repetitions recorded for each stimuli. Response duration was 100 ms.

**Obtaining equivalence constraints over stimuli pairs**   The distances between responses were measured using a normalized $\chi^2$ distance measure. All responses to both stimuli (40 responses in total) were superimposed to generate a single high-resolution PSTH. Then, this PSTH was non-uniformly binned so that each bin contained at least 10 spikes. The same bins were then used to generate the PSTHs of the responses to the two stimuli separately. For similar responses, we would expect that on average each bin in these histograms would contain 5 spikes. Formally, let $N$ denote the number of bins in each histogram, and let $r_1^i$, $r_2^i$ denote the number of spikes in the $i$'th bin in each of the two histograms respectively. The distance between pairs of histograms is given by: $\chi^2(r_1^i, r_2^i) = \sum_{i=1}^{N} \frac{(r_1^i - r_2^i)^2}{(r_1^i + r_2^i)/2} / (N-1)$.

In order to identify pairs (or small groups) of similar responses, we computed the normalized $\chi^2$ distance matrix over all pairs of responses, and used the complete-linkage algorithm to cluster the responses into $8-12$ clusters. All of the points in each cluster were marked as similar to one another, thus providing positive equivalence constraints. In order to obtain negative equivalence constraints, for each cluster $c_i$ we used the $2-3$ furthest clusters from it to define negative constraints. All pairs, composed of a point from cluster $c_i$ and another point from these distant clusters, were used as negative constraints.

**Distance learning method**   In this paper, we use the *DistBoost* algorithm [7], which is a semi-supervised boosting learning algorithm that learns a distance function using unlabeled datapoints and equivalence constraints. The algorithm boosts weak learners which are soft partitions of the input space, that are computed using the constrained Expectation-Maximization (cEM) algorithm [15]. The *DistBoost* algorithm, which is briefly summarized in  1, has been previously used in several different applications and has been shown to perform well [7, 17].

**Evaluation methods**   In order to evaluate the quality of the learned distance function, we measured the correlation between the distances computed by our distance learning algorithm to those induced by the $\chi^2$ distance over the responses. For each stimulus we measured the distances to all other stimuli using the learnt distance function. We then computed the rank-order (Spearman) correlation coefficient between these learnt distances in the stimulus domain and the $\chi^2$ distances between the appropriate responses. This procedure produced a single correlation coefficient for each of the 32 stimuli, and the average correlation coefficient across all stimuli was used as the overall performance measure.

**Parameter selection**   The following parameters of the *DistBoost* algorithm can be fine-tuned: (1) the input dimensionality $d = $ 21-30, (2) the number of Gaussian models in each weak learner $M = $ 2-4, (3) the number of clusters used to extract equivalence constraints $C = $ 8-12, and (4) the number of distant clusters used to define negative constraints $numAnti = $ 2-3. Optimal parameters were determined separately for each of the 22 cells, based *solely* on the training data. Specifically, in the cross-validation testing we used a validation paradigm: Using the 31 training stimuli, we removed an additional datapoint and trained our algorithm on the remaining 30 points. We then validated its performance using the left out datapoint. The optimal cell specific parameters were determined using this approach.

**Algorithm 1** The *DistBoost* Algorithm

---

**Input**:

  **Data points:** $(x_1, ..., x_n)$, $x_k \in \mathcal{X}$

  **A set of equivalence constraints:** $(x_{i_1}, x_{i_2}, y_i)$, where $y_i \in \{-1, 1\}$

  **Unlabeled pairs of points:** $(x_{i_1}, x_{i_2}, y_i = *)$, implicitly defined by all unconstrained pairs of points

- Initialize $W_{i_1 i_2}^1 = 1/(n^2)$   $i_1, i_2 = 1, \ldots, n$ (weights over pairs of points)
  $$w_k = 1/n \qquad k = 1, \ldots, n \text{ (weights over data points)}$$

- For $t = 1, .., T$

  1. Fit a constrained GMM (weak learner) on weighted data points in $\mathcal{X}$ using the equivalence constraints.

  2. Generate a weak hypothesis $\tilde{h}_t : \mathcal{X} \times \mathcal{X} \rightarrow [-1, 1]$ and define a weak distance function as $h_t(x_i, x_j) = \frac{1}{2}\left(1 - \tilde{h}_t(x_i, x_j)\right) \in [0, 1]$

  3. Compute $r_t = \sum\limits_{(x_{i_1}, x_{i_2}, y_i = \pm 1)} W_{i_1 i_2}^t y_i h_t(x_{i_1}, x_{i_2})$, only over **labeled** pairs. Accept the current hypothesis only if $r_t > 0$.

  4. Choose the hypothesis weight $\alpha_t = \frac{1}{2}\ln(\frac{1+r_t}{1-r_t})$

  5. Update the weights of **all** points in $\mathcal{X} \times \mathcal{X}$ as follows:
  $$W_{i_1 i_2}^{t+1} = \begin{cases} W_{i_1 i_2}^t \exp(-\alpha_t y_i \tilde{h}_t(x_{i_1}, x_{i_2})) & y_i \in \{-1, 1\} \\ W_{i_1 i_2}^t \exp(-\alpha_t) & y_i = * \end{cases}$$

  6. Normalize: $W_{i_1 i_2}^{t+1} = \dfrac{W_{i_1 i_2}^{t+1}}{\sum\limits_{i_1, i_2 = 1}^{n} W_{i_1 i_2}^{t+1}}$

  7. Translate the weights from $\mathcal{X} \times \mathcal{X}$ to $\mathcal{X}$: $w_k^{t+1} = \sum_j W_{kj}^{t+1}$

**Output**: A final distance function $\mathcal{D}(x_i, x_j) = \sum_{t=1}^{T} \alpha_t h_t(x_i, x_j)$

---

# 4 Results

**Cell-specific distance functions**   We begin our analysis with an evaluation of the fitting power of the method, by training with the entire set of 32 stimuli (see Fig. 1). In general almost all of the correlation values are positive and they are quite high. The average correlation over all cells is $0.58$ with $ste = 0.023$.

In order to evaluate the generalization potential of our approach, we used a Leave-One-Out (LOU) cross-validation paradigm. In each run, we removed a single stimulus from the dataset, trained our algorithm on the remaining 31 stimuli, and then tested its performance on the datapoint that was left out (see Fig. 3). In each histogram we plot the test correlations of a single cell, obtained when using the LOU paradigm over all of the 32 stimuli. As can be seen, on some cells our algorithm obtains correlations that are as high as $0.41$, while for other cells the average test correlation is less then $0.1$. The average correlation over all cells is $0.26$ with $ste = 0.019$.

Not surprisingly, the train results (Fig. 1) are better than the test results (Fig. 3). Interestingly, however, we found that there was a significant correlation between the training performance and the test performance $C = 0.57, p < 0.05$ (see Fig. 2, left).

**Boosting the performance of weak cells**   In order to boost the performance of cells with low average correlations, we constructed the following experiment: We clustered the responses of each cell, using the complete-linkage algorithm over the $\chi^2$ distances with 4 clusters. We then used the $F_{\frac{1}{2}}$ score that evaluates how well two clustering partitions are in agreement with one another ($F_{\frac{1}{2}} = \frac{2*P*R}{P+R}$, where $P$ denotes precision and $R$ denotes recall.). This measure was used to identify pairs of cells whose partition of the stimuli was most similar to each other. In our experiment we took the four cells with the lowest

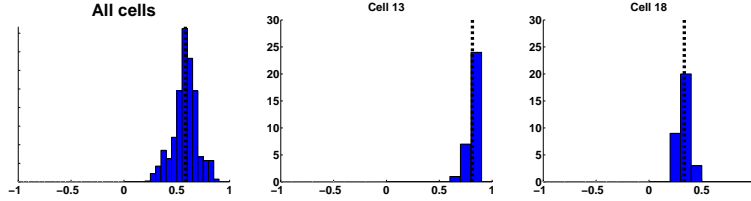

Figure 1: Left: Histogram of train rank-order correlations on the entire ensemble of cells. The rank-order correlations were computed between the learnt distances and the distances between the recorded responses for each single stimulus ($N = 22 * 32$). Center: train correlations for a "strong" cell. Right: train correlations for a "weak" cell. Dotted lines represent average values.

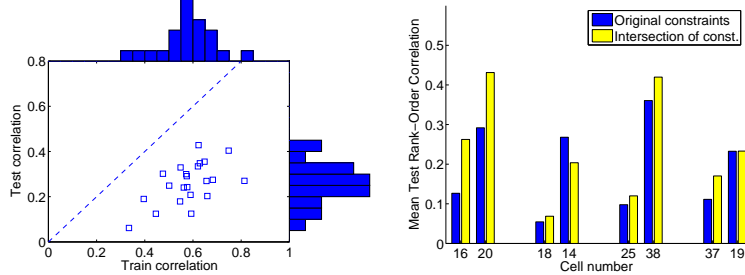

Figure 2: Left: Train vs. test cell specific correlations. Each point marks the average correlation of a single cell. The correlation between train and test is $0.57$ with $p = 0.05$. The distribution of train and test correlations is displayed as histograms on the top and on the right respectively. Right: Test rank-order correlations when training using constraints extracted from each cell separately, and when using the intersection of the constraints extracted from a pair of cells. This procedure always improves the performance of the weaker cell, and usually also improves the performance of the stronger cell

performance (right column of Fig 3), and for each of them used the $F_{\frac{1}{2}}$ score to retrieve the most similar cell. For each of these pairs, we trained our algorithm once more, using the constraints obtained by intersecting the constraints derived from the two cells in the pair, in the LOU paradigm. The results can be seen on the right plot in Fig 2. On all four cells, this procedure improved LOUT test results. Interestingly and counter-intuitively, when training the better performing cell in each pair using the intersection of its constraints with those from the poorly performing cell, results deteriorated only for one of the four better performing cells.

**Stimulus classification** The cross-validation results induced a partition of the stimulus space into narrowband and wideband stimuli. We measured the predictability of each stimulus by averaging the LOU test results obtained for the stimulus across all cells (see Fig. 4). Our analysis shows that wideband stimuli are more predictable than narrowband stimuli, despite the fact that the neuronal responses to these two groups are not different as a whole. Whereas the non-linearity in the interactions between narrowband and wideband stimuli has already been noted before [9], here we further refine this observation by demonstrating a significant difference between the behavior of narrow and wideband stimuli with respect to the predictability of the similarity between their responses.

## 5  Discussion

In the standard approach to auditory modeling, a linear or weakly non-linear model is fitted to the data, and neuronal properties are read from the resulting model. The usefulness of this approach is limited however by the weak predictability of A1 responses when using such models. In order to overcome this limitation, we reformulated the problem of char-

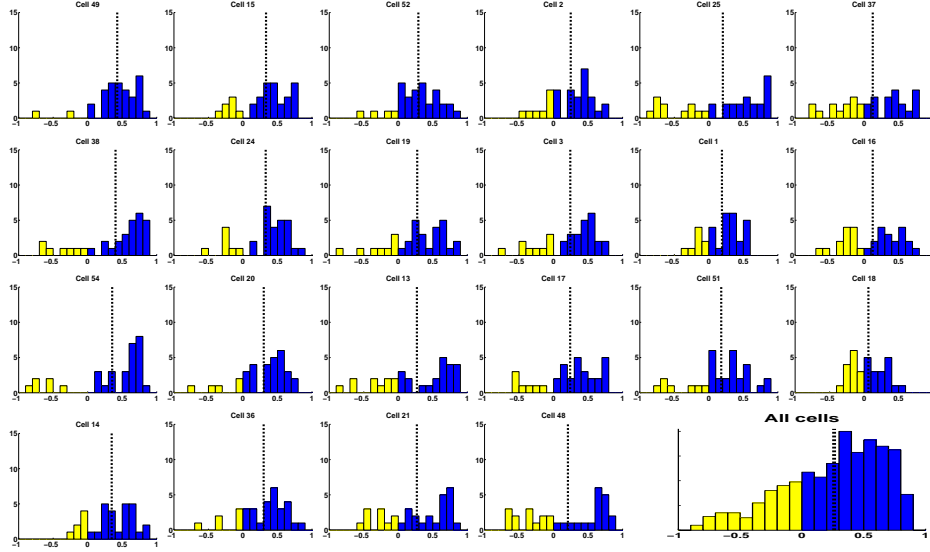

Figure 3: Histograms of cell specific test rank-order correlations for the 22 cells in the dataset. The rank-order correlations compare the predicted distances to the distances between the recorded responses, measured on a single stimulus which was left out during the training stage. For visualization purposes, cells are ordered (columns) by their average test correlation per stimulus in descending order. Negative correlations are in yellow, positive in blue.

acterizing neuronal responses of highly non-linear neurons. We use the neural data as a guide for training a highly non-linear distance function on stimulus space, which is compatible with the neural responses. The main result of this paper is the demonstration of the feasibility of this approach.

Two further results underscore the usefulness of the new formulation. First, we demonstrated that we can improve the test performance of a distance function by using constraints on the similarity or dissimilarity between stimuli derived from the responses of multiple neurons. Whereas we expected this manipulation to improve the test performance of the algorithm on the responses of neurons that were initially poorly predicted, we found that it actually improved the performance of the algorithm also on neurons that were rather well predicted, although we paired them with neurons that were poorly predicted. Thus, it is possible that intersecting constraints derived from multiple neurons uncover regularities that are hard to extract from individual neurons.

Second, it turned out that some stimuli consistently behaved better than others across the neuronal population. This difference was correlated with the acoustic structure of the stimuli: those stimuli that contained the weak background component (wideband stimuli) were generally predicted better. This result is surprising both because background component is substantially weaker than the other acoustic components in the stimuli (by as much as 35-40 dB). It may mean that the relationships between physical structure (as characterized by the Cepstral parameters) and the neuronal responses becomes simpler in the presence of the background component, but is much more idiosyncratic when this component is absent. This result underscores the importance of interactions between narrow and wideband stimuli for understanding the complexity of cortical processing.

The algorithm is fast enough to be used in near real-time. It can therefore be used to guide real experiments. One major problem during an experiment is that of stimulus selection: choosing the best set of stimuli for characterizing the responses of a neuron. The distance functions trained here can be used to direct this process. For example, they can be used to

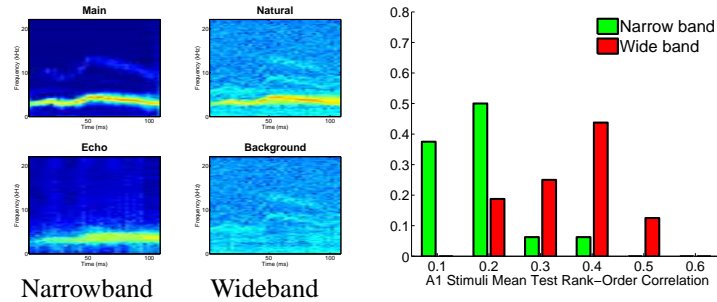

Figure 4: Left: spectrograms of input stimuli, which are four different versions of a single natural bird chirp. Right: Stimuli specific correlation values averaged over the entire ensemble of cells. The predictability of wideband stimuli is clearly better than that of the narrowband stimuli.

find surprising stimuli: either stimuli that are very different in terms of physical structure but that would result in responses that are similar to those already measured, or stimuli that are very similar to already tested stimuli but that are predicted to give rise to very different responses.

# References

[1] O. Bar-Yosef, Y. Rotman, and I. Nelken. Responses of Neurons in Cat Primary Auditory Cortex to Bird Chirps: Effects of Temporal and Spectral Context. *J. Neurosci.*, 22(19):8619–8632, 2002.

[2] D. T. Blake and M. M. Merzenich. Changes of AI Receptive Fields With Sound Density. *J Neurophysiol*, 88(6):3409–3420, 2002.

[3] G. Chechik, A. Globerson, M.J. Anderson, E.D. Young, I. Nelken, and N. Tishby. Group redundancy measures reveal redundancy reduction in the auditory pathway. In *NIPS*, 2002.

[4] R. C. deCharms, D. T. Blake, and M. M. Merzenich. Optimizing Sound Features for Cortical Neurons. *Science*, 280(5368):1439–1444, 1998.

[5] D. A. Depireux, J. Z. Simon, D. J. Klein, and S. A. Shamma. Spectro-Temporal Response Field Characterization With Dynamic Ripples in Ferret Primary Auditory Cortex. *J Neurophysiol*, 85(3):1220–1234, 2001.

[6] J. J. Eggermont, P. M. Johannesma, and A. M. Aertsen. Reverse-correlation methods in auditory research. *Q Rev Biophys.*, 16(3):341–414, 1983.

[7] T. Hertz, A. Bar-Hillel, and D. Weinshall. Boosting margin based distance functions for clustering. In *ICML*, 2004.

[8] N. Kowalski, D. A. Depireux, and S. A. Shamma. Analysis of dynamic spectra in ferret primary auditory cortex. I. Characteristics of single-unit responses to moving ripple spectra. *J Neurophysiol*, 76(5):3503–3523, 1996.

[9] L. Las, E. A. Stern, and I. Nelken. Representation of Tone in Fluctuating Maskers in the Ascending Auditory System. *J. Neurosci.*, 25(6):1503–1513, 2005.

[10] C. K. Machens, M. S. Wehr, and A. M. Zador. Linearity of Cortical Receptive Fields Measured with Natural Sounds. *J. Neurosci.*, 24(5):1089–1100, 2004.

[11] L. M. Miller, M. A. Escabi, H. L. Read, and C. E. Schreiner. Spectrotemporal Receptive Fields in the Lemniscal Auditory Thalamus and Cortex. *J Neurophysiol*, 87(1):516–527, 2002.

[12] I. Nelken. Processing of complex stimuli and natural scenes in the auditory cortex. *Current Opinion in Neurobiology*, 14(4):474–480, 2004.

[13] Y. Rotman, O. Bar-Yosef, and I. Nelken. Relating cluster and population responses to natural sounds and tonal stimuli in cat primary auditory cortex. *Hearing Research*, 152(1-2):110–127, 2001.

[14] M. Sahani and J. F. Linden. How linear are auditory cortical responses? In *NIPS*, 2003.

[15] N. Shental, A. Bar-Hilel, T. Hertz, and D. Weinshall. Computing Gaussian mixture models with EM using equivalence constraints. In *NIPS*, 2003.

[16] F. E. Theunissen, K. Sen, and A. J. Doupe. Spectral-Temporal Receptive Fields of Nonlinear Auditory Neurons Obtained Using Natural Sounds. *J. Neurosci.*, 20(6):2315–2331, 2000.

[17] C. Yanover and T. Hertz. Predicting protein-peptide binding affinity by learning peptide-peptide distance functions. In *RECOMB*, 2005.
